# Bayesian Sparse Factor Models and DAGs Inference and Comparison

**Ricardo Henao**
DTU Informatics
Technical University of Denmark
2800 Lyngby, Denmark
Bioinformatics Centre
University of Copenhagen
2200 Copenhagen, Denmark
rhenao@binf.ku.dk

**Ole Winther**
DTU Informatics
Technical University of Denmark
2800 Lyngby, Denmark
Bioinformatics Centre
University of Copenhagen
2200 Copenhagen, Denmark
owi@imm.dtu.dk

## Abstract

In this paper we present a novel approach to learn directed acyclic graphs (DAGs) and factor models within the same framework while also allowing for model comparison between them. For this purpose, we exploit the connection between factor models and DAGs to propose Bayesian hierarchies based on spike and slab priors to promote sparsity, heavy-tailed priors to ensure identifiability and predictive densities to perform the model comparison. We require identifiability to be able to produce variable orderings leading to valid DAGs and sparsity to learn the structures. The effectiveness of our approach is demonstrated through extensive experiments on artificial and biological data showing that our approach outperform a number of state of the art methods.

## 1  Introduction

Sparse factor models have proven to be a very versatile tool for detailed modeling and interpretation of multivariate data, for example in the context of gene expression data analysis [1, 2]. A sparse factor model encodes the prior knowledge that the latent factors only affect a limited number of the observed variables. An alternative way of modeling the data is through linear regression between the measured quantities. This multiple regression model is a well-defined multivariate probabilistic model if the connectivity (non-zero weights) defines a directed acyclic graph (DAG). What usually is done in practice is to consider either factor or DAG models. Modeling the data with both types of models at the same time and then perform model comparison should provide additional insight as these models are complementary and often closely related. Unfortunately, existing off-the-shelf models are specified in such a way that makes direct comparison difficult. A more principled idea that can phrased in Bayesian terms is for example to find an equivalence between both models, then represent them using a common/comparable hierarchy, and finally use a marginal likelihood or a predictive density to select one of them. Although a formal connection between factor models and DAGs has been already established in [3], this paper makes important extensions such as explicitly modeling sparsity, stochastic search over the order of the variables and model comparison.

Is well known that learning the structure of graphical models, in particular DAGs is a very difficult task because it turns out to be a combinatorial optimization problem known to be NP-hard [4]. A commonly used approach for structure learning is to split the problem into two stages using the fact that the space of variable orderings is far more smaller than the space of all possible structures, e.g. by first attempting to learn a suitable permutation of the variables and then the skeleton of the structure given the already found ordering or viceversa. Most of the work so far for continuous data assumes linearity and Gaussian variables hence they can only recover the DAG structure up

to Markov equivalence [5, 6, 7, 8], which means that some subset of links can be reversed without changing the likelihood [9]. To break the Markov equivalence usually experimental (interventional) data in addition to the observational (non-interventional) data is required [10]. In order to obtain identifiability from purely observational data, strong assumptions have to to be made [11, 3, 12]. In this work we follow the line of [3] by starting from a linear factor model and ensure identifiability by using non-normal heavy-tailed latent variables. As a byproduct we find a set of candidate orderings compatible with a linear DAG, i.e. a mixing matrix which is "close to" triangular. Finally, we may perform model comparison between the factor and DAG models inferred with fixed orderings taken from the candidate set.

The rest of the paper is organized as follows. Sections 2 to 5 we motivate and describe the different ingredients in our method, in Section 6 we discuss existing work, in Section 7 experiments on both artificial and real data are presented, and Section 8 concludes with a discussion and perspectives for future work.

## 2 From DAGs to factor models

We will assume that an ordered $d$-dimensional data vector $\mathbf{Px}$ can be represented as a directed acyclic graph with only observed nodes, where $\mathbf{P}$ is the usually unknown true permutation matrix. We will focus entirely on linear models such that the value of each variable is a linear weight combination of parent nodes plus a driving signal $\mathbf{z}$

$$\mathbf{x} = \mathbf{P}^{-1}\mathbf{BPx} + \mathbf{z} \,, \tag{1}$$

where $\mathbf{B}$ is a strictly lower triangular square matrix. In this setting, each non-zero element of $\mathbf{B}$ corresponds to a link in the DAG. Solving for $\mathbf{x}$ we can rewrite the problem as

$$\mathbf{x} = \mathbf{P}^{-1}\mathbf{APz} = \mathbf{P}^{-1}(\mathbf{I} - \mathbf{B})^{-1}\mathbf{Pz} \,, \tag{2}$$

which corresponds to a noise-free linear factor model with the restriction that $\mathbf{P}^{-1}\mathbf{AP}$ must have a sparsity pattern that can be permuted to a triangular form since $(\mathbf{I}-\mathbf{B})^{-1}$ is triangular. This requirement alone is not enough to ensure identifiability (up to scaling and permutation of columns $\mathbf{P}_f$)[1]. We further have to use prior knowledge about the distribution of the factors $\mathbf{z}$. A necessary condition is that these must be a set of non-Gaussian independent variables [11]. For heavy-tailed data is it often sufficient in practice to use a model with heavier tails than Gaussian [13]. If the requirements for $\mathbf{A}$ and for the distribution of $\mathbf{z}$ are met, we can first estimate $\mathbf{P}^{-1}\mathbf{AP}$ and subsequently find $\mathbf{P}$ searching over the space of all possible orderings. Recently, [3] applied the fastICA algorithm to solve for the inverse mixing matrix $\mathbf{P}^{-1}\mathbf{A}^{-1}\mathbf{P}$. To find a candidate solution for $\mathbf{B}$, $\mathbf{P}$ is set such that $\mathbf{B}$ found from the direct relation equation (1), $\mathbf{B} = \mathbf{I} - \mathbf{A}^{-1}$ (according to magnitude-based criterion) is as close as possible to lower triangular. In the final step the Wald statistic is used for pruning $\mathbf{B}$ and the chi-square test is used for model selection.

In our work we also exploit the relation between the factor models and linear DAGs. We apply a Bayesian approach to learning a sparse factor models and DAGs, and the stochastic search for $\mathbf{P}$ is performed as an integrated part of inference of the sparse factor model. The inference of factor model (including order) and DAG parameters are performed as two separate inferences such that the only input that comes from the first part is a set of candidate orders.

## 3 From factor models to DAGs

Our first goal is to perform model inference in the families of factor and linear DAG models. We specify the joint distribution or *probability of everything*, e.g. for the factor model, as

$$p(\mathbf{X}, \mathbf{A}, \mathbf{Z}, \mathbf{\Psi}, \mathbf{P}, \cdot) = p(\mathbf{X}|\mathbf{A}, \mathbf{Z}, \mathbf{P}, \cdot)p(\mathbf{A}|\cdot)p(\mathbf{Z}|\cdot)p(\mathbf{\Psi}|\cdot)p(\mathbf{P}|\cdot)p(\cdot) \,,$$

where $\mathbf{X} = [\mathbf{x}_1, \ldots, \mathbf{x}_N]$, $\mathbf{Z} = [\mathbf{z}_1, \ldots \mathbf{z}_N]$, $N$ is the number of observations and $(\cdot)$ indicates additional parameters in the hierarchical models. The prior over permutation $p(\mathbf{P}|\cdot)$ will always be chosen to be uniform over the $d!$ possible values. The actual sampling based inference for $\mathbf{P}$ is discussed in the next section and the standard Gibbs sampling components are provided in the supplementary material. Model comparison should ideally be performed using the marginal likelihood. This is more difficult to calculate with sampling than obtaining samples from the posterior so we use the predictive densities on a test set as a yardstick.

**Factor model** Instead of using the noise-free factor model of equation (2) we allow for additive noise $\mathbf{x} = \mathbf{P}_r^{-1}\mathbf{A}\mathbf{P}_c\mathbf{z} + \boldsymbol{\epsilon}$, where $\boldsymbol{\epsilon}$ is an additional Gaussian noise term with diagonal covariance matrix $\boldsymbol{\Psi}$, i.e. uncorrelated noise, to account for independent measurement noise, $\mathbf{P}_r = \mathbf{P}$ is the permutation matrix for the rows of $\mathbf{A}$ and $\mathbf{P}_c = \mathbf{P}_f\mathbf{P}_r$ another permutation for the columns with $\mathbf{P}_f$ accounting for the permutation freedom of the factors. We will not restrict the mixing matrix $\mathbf{A}$ to be triangular. Instead we infer $\mathbf{P}_r$ and $\mathbf{P}_c$ using a stochastic search based upon closeness to triangular as measured by a masked likelihood, see below. Now we can specify a hierarchy for the Bayesian model as follows

$$\mathbf{X}|\mathbf{P}_r,\mathbf{A},\mathbf{P}_c,\mathbf{Z},\boldsymbol{\Psi} \sim \mathcal{N}(\mathbf{X}|\mathbf{P}_r^{-1}\mathbf{A}\mathbf{P}_c\mathbf{Z},\boldsymbol{\Psi})\,, \qquad \mathbf{Z} \sim \pi(\mathbf{Z}|\cdot)\,,$$
$$\psi_i^{-1}|s_s,s_r \sim \mathrm{Gamma}(\psi_i^{-1}|s_s,s_r)\,, \qquad \mathbf{A} \sim \rho(\mathbf{A}|\cdot)\,, \tag{3}$$

where $\psi_i$ are elements of $\boldsymbol{\Psi}$. For convenience, to exploit conjugate exponential families we are placing a gamma prior on the precision of $\boldsymbol{\epsilon}$ with shape $s_s$ and rate $s_r$. Given that the data is standardized, the selection of hyperparameters for $\psi_i$ is not very critical as long as both "signal and noise" are supported. The prior should favor small values of $\psi_i$ as well as providing support for $\psi_i = 1$ such that certain variables can be explained solely by noise (we set $s_s = 2$ and $s_r = 0.05$ in the experiments).

For the factors we use a heavy-tailed prior $\pi(\mathbf{Z}|\cdot)$ in the form of a Laplace distribution parameterized for convenience as a scale mixture of Gaussians [14]

$$z_{jn}|\mu,\lambda \sim \mathrm{Laplace}(z_{jn}|\mu,\lambda) = \int_0^\infty \mathcal{N}(z_{jn}|\mu,\upsilon)\mathrm{Exponential}(\upsilon_{jn}|\lambda^2)d\upsilon_{jn}\,, \tag{4}$$
$$\lambda^2|\ell_s,\ell_r \sim \mathrm{Gamma}(\lambda^2|\ell_s,\ell_r)\,, \tag{5}$$

where $z_{jn}$ is an element of $\mathbf{Z}$, $\lambda$ is the rate and $\upsilon$ has an exponential distribution acting as mixing density. Furthermore, we place a gamma distribution on $\lambda^2$ to get conditionals for $\upsilon$ and $\lambda^2$ in standard conjugate families. We let the components of $\mathbf{Z}$ have on average unit variance. This is achieved by setting $\ell_s/\ell_r = 2$ (we set $\ell_s = 4$ and $\ell_r = 2$). Alternatively one may use a $t$ distribution—again as scale mixture of Gaussians—which can to interpolate between very heavy-tailed (power law) and very light tails, i.e. becoming Gaussian when degrees of freedom approaches infinity. However such flexibility comes at the price of being more difficult to select its hyperparameters, because the model could become unidentified for some settings.

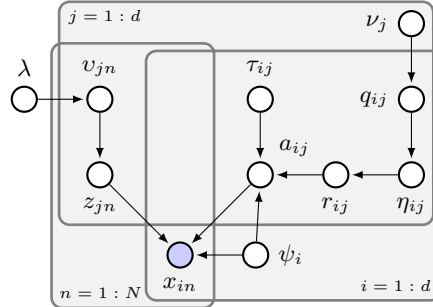

Figure 1: Graphical model for Bayesian hierarchy in equation (3).

The prior $\rho(\mathbf{A}|\cdot)$ for the mixing matrix should be biased towards sparsity because we want to infer something close to a triangular matrix. Here we adopt a two-layer discrete spike and slab prior for the elements $a_{ij}$ of $\mathbf{A}$ similar to the one in [2]. The first layer in the prior control the sparsity of each element $a_{ij}$ individually, whereas the second layer impose a per-factor sparsity level to allow elements within the same factor to share information. The hierarchy can be written as

$$a_{ij}|r_{ij},\psi_i,\tau_{ij} \sim (1-r_{ij})\delta(a_{ij}) + r_{ij}\mathcal{N}(a_{ij}|0,\psi_i\tau_{ij})\,,$$
$$\tau_{ij}^{-1}|t_s,t_r \sim \mathrm{Gamma}(\tau_{ij}^{-1}|t_s,t_r)\,,$$
$$r_{ij}|\eta_{ij} \sim \mathrm{Bernoulli}(r_{ij}|\eta_{ij})\,,$$
$$\eta_{ij}|q_{ij},\alpha_p,\alpha_m \sim (1-q_{ij})\delta(\eta_{ij}) + q_{ij}\mathrm{Beta}(\eta_{ij}|\alpha_p\alpha_m,\alpha_p(1-\alpha_m))\,, \tag{6}$$
$$q_{ij}|\nu_j \sim \mathrm{Bernoulli}(q_{ij}|\nu_j)\,,$$
$$\nu_j|\beta_m,\beta_p \sim \mathrm{Beta}(\nu_j|\beta_p\beta_m,\beta_p(1-\beta_m))\,,$$

where $\delta(\cdot)$ is a Dirac $\delta$-function. The prior above specify a point mass mixture over $a_{ij}$ with mask $r_{ij}$. The expected probability of $a_{ij}$ to be non-zero is $\eta_{ij}$ and is controlled through a beta hyperprior with mean $\alpha_m$ and precision $\alpha_p$. Besides, each factor has a common sparsity rate $\nu_j$ that let the elements $\eta_{ij}$ to be exactly zero with probability $1 - \nu_j$ through a beta distribution with mean $\beta_m$ and

precision $\beta_p$, turning the distribution of $\eta_{ij}$ bimodal over the unit interval. The magnitude of non-zero elements in $\mathbf{A}$ is specified through the slab distribution depending on $\tau_{ij}$. The parameters for $\tau_{ij}$ should be specified in the same fashion as $\psi_i$ but putting more probability mass around $a_{ij} = 1$, for instance $t_s = 4$ and $t_r = 10$. Note that we scale the variances with $\psi_i$ since it makes the model easier to specify and tend to have better mixing properties [15]. The masking matrix $r_{ij}$ with parameters $\eta_{ij}$ should be somewhat diffuse while favoring relatively large masking probabilities, e.g. $\alpha_p = 10$ and $\alpha_m = 0.9$. Additionally, $q_j$ and should favor very small values with low variance, this is for example $\beta_p = 1000$ and $\beta_m = 0.005$. The graphical model for the entire hierarchy in (3) omitting parameters is shown in Figure 1.

**DAG**   We make the following Bayesian specification of linear DAG model of equation (1) as

$$\mathbf{X}|\mathbf{P}_r, \mathbf{B}, \mathbf{X}, \cdot \sim \pi(\mathbf{X} - \mathbf{P}_r^{-1}\mathbf{B}|\cdot), \quad \mathbf{B} \sim \rho(\mathbf{B}|\cdot), \tag{7}$$

where $\pi$ and $\rho$ are given by equations (4) and (6). The Bayesian specification for the DAG has a similar graphical model to the one in Figure 1 but without noise variances $\boldsymbol{\Psi}$. The factor model needs only shared variance parameter $\lambda$ for the Laplace distributed $z_{jn}$ because a change of scale in $\mathbf{A}$ is equivalent to change of variance in $z_{jn}$. The DAG on the other hand, needs individual variance parameters because it has no scaling freedom. Given that we know that $\mathbf{B}$ is strictly lower triangular, it should be in general less sparse than $\mathbf{A}$, thus we use a different setting for the sparsity prior, i.e. $\beta_p = 100$ and $\beta_m = 0.01$.

## 4   Sampling based inference

For given permutation $\mathbf{P}$, Gibbs sampling can be used for inference of the remaining parameters. Details of Gibbs sampler is given in the supplementary material and we will focus on the non-standard inference corresponding to the sampling over permutations. There are basically two approaches to find $\mathbf{P}$, one is perform the inference for parameters and $\mathbf{P}$ jointly with $\mathbf{B}$ restricted to be triangular. The other is to let the factor model be unrestricted and search for $\mathbf{P}$ according to a criterion that does not affect parameter inference. Here we prefer the latter for two reasons. First, joint combinatorial and parameter inference in this model will probably have poor mixing with slow convergence. Second, we are also interested in comparing the factor model against the DAG for cases when we cannot really assume that the data is well approximated by a DAG. In our approach the proposal $\mathbf{P}^\star$ corresponds to picking two of the elements in the order vector by random and exchanging them. Other approaches such as restricting to pick two adjacent elements have been suggested as well [16, 7]. For the linear DAG model we are not performing joint inference of $\mathbf{P}$ and the model parameters. Rather we use a set of $\mathbf{P}$s found for the factor model to be good candidates for the DAG.

The stochastic search for $\mathbf{P} = \mathbf{P}_c$ goes as follows: we make inference for the unrestricted factor model, propose $\mathbf{P}_r^\star$ and $\mathbf{P}_c^\star$ independently according $q(\mathbf{P}_r^\star|\mathbf{P}_r)q(\mathbf{P}_c^\star|\mathbf{P}_c)$ which is the uniform two variable random exchange. With this proposal and the flat prior over $\mathbf{P}$, we use a Metropolis-Hastings acceptance probability simply as the ratio of likelihoods with $\mathbf{A}$ *masked to have zeros above its diagonal* (through masking matrix $\mathbf{M}$)

$$\xi_{\to\star} = \frac{\mathcal{N}(\mathbf{X}|(\mathbf{P}_r^\star)^{-1}(\mathbf{M} \odot \mathbf{P}_r^\star \mathbf{A}(\mathbf{P}_c^\star)^{-1})\mathbf{P}_c^\star, \boldsymbol{\Psi})}{\mathcal{N}(\mathbf{X}|\mathbf{P}_r^{-1}(\mathbf{M} \odot \mathbf{P}_r \mathbf{A}\mathbf{P}_c^{-1})\mathbf{P}_c, \boldsymbol{\Psi})},$$

The procedure can be seen as a simple approach for generating hypotheses about good, close to triangular $\mathbf{A}$, orderings in a model where the spike and slab prior provides bias towards sparsity.

To learn DAGs we first perform inference on the factor model specified by the hierarchy in (3) to obtain a set of ordering candidates sorted according to their usage during sampling—after the burn-in period. It is possible that the estimation of $\mathbf{A}$ might contain errors, e.g. a false zero entry on $\mathbf{A}$ allowing several orderings leading to several lower triangular versions of $\mathbf{A}$, only one of those being actually correct. Thus, we propose not only to use the best candidate but a set of top candidates of size $m_{\text{top}} = 10$. Then we perform inference on the DAG model corresponding to the structure search hierarchy in (7), for each one of the permutation candidates being considered, $\mathbf{P}_r^{(1)}, \ldots, \mathbf{P}_r^{(m_{\text{top}})}$. Finally, we select the DAG model among candidates using the predictive distribution for the DAG when a test set is available or just the likelihood if not.

# 5 Predictive distributions and model comparison

Given that our model produces both DAG and a factor model estimates at the same time, it could be interesting to estimate also whether one option is better than the other given the observed data, for example in exploratory analysis when the DAG assumption is just one reasonable option. In order to perform the model comparison, we use predictive densities $p(\mathbf{X}^\star|\mathbf{X}, \mathcal{M})$ with $\mathcal{M} = \{\mathcal{M}_{\mathrm{FA}}, \mathcal{M}_{\mathrm{DAG}}\}$, instead of marginal likelihoods because the latter is difficult and expensive to compute by sampling, requiring for example thermodynamic integration. With Gibbs sampling, we draw samples from the posterior distributions $p(\mathbf{A}, \boldsymbol{\Psi}, \lambda|\mathbf{X}, \cdot)$ and $p(\mathbf{B}, \lambda_1, \ldots, \lambda_m|\mathbf{X}, \cdot)$. The average over the extensive variables associated with the test points $p(\mathbf{Z}^\star|\cdot)$ is a bit more complicated because naively drawing samples from $p(\mathbf{Z}^\star|\cdot)$ gives an estimator with high variance—for $\psi_i \ll \upsilon_{jn}$. In the following we describe how to do it for each model, omitting the permutation matrices for clarity.

**Factor model**   We can compute the predictive distribution by taking the likelihood in equation (3) and marginalizing $\mathbf{Z}$. Since the integral has no closed form we can approximate it using the Gaussian distribution from the scale mixture representation as

$$p(\mathbf{X}^\star|\mathbf{A}, \boldsymbol{\Psi}, \cdot) = \int p(\mathbf{X}^\star|\mathbf{A}, \mathbf{Z}, \boldsymbol{\Psi}) p(\mathbf{Z}|\cdot) d\mathbf{Z} \approx \frac{1}{\mathrm{rep}} \prod_n \sum_r^{\mathrm{rep}} \mathcal{N}(\mathbf{x}_n^\star|\mathbf{0}, \mathbf{A}^\top \mathbf{U}_n \mathbf{A} + \boldsymbol{\Psi}) \,,$$

where $\mathbf{U}_n = \mathrm{diag}(\upsilon_{1n}, \ldots, \upsilon_{dn})$, the $\upsilon_{jn}$ are sampled from the prior and $\mathrm{rep}$ is the number of samples generated to approximate the intractable integral ($\mathrm{rep} = 500$ in the experiments). Then we can average over $p(\mathbf{A}, \boldsymbol{\Psi}, \lambda|\mathbf{X}, \cdot)$ to obtain $p(\mathbf{X}^\star|\mathbf{X}, \mathcal{M}_{\mathrm{FA}})$.

**DAG**   In this case the predictive distribution is rather easy because the marginal over $\mathbf{Z}$ in equation (4) is just a Laplace distribution with mean $\mathbf{BX}$

$$p(\mathbf{X}^\star|\mathbf{B}, \cdot) = \int p(\mathbf{X}^\star|\mathbf{B}, \mathbf{X}, \mathbf{Z}) p(\mathbf{Z}|\cdot) d\mathbf{Z} = \prod_{i,n} \mathrm{Laplace}(x_{ij}|[\mathbf{BX}]_{in}, \lambda_i) \,,$$

where $[\mathbf{BX}]_{ij}$ is the element indexed by the $i$-th row and $n$-th column of $\mathbf{BX}$. In practice we compute the predictive densities for a particular $\mathbf{X}^\star$ during sampling and then select the model based on its ratio. Note that both predictive distributions depend directly on $\lambda$—the rate of Laplace distribution, making the estimates highly dependent on its value. This is why it is important to have the hyperprior on $\lambda$ of equation (5) instead of just fixing its value.

# 6 Existing work

Among the existing approaches to DAG learning, our work is most closely related to LiNGAM (Linear Non-Gaussian Acyclic Model for causal discovery) [3] with several important differences: Since LiNGAM relies on fastICA to learn the mixing is not inherently sparse, hence a pruning procedure based on Wald statistic and model fit second order information should be applied after obtaining an ordering for the variables. The order search in LiNGAM assumes that there is not estimation errors during fastICA model inference, then a single ordering candidate is produced. LiNGAM produces and select a final model among several candidates, but in contrast to our method such candidates are not different DAGs with different variable orderings but DAGs with different sparsity levels. The factor model inference in LiNGAM, namely fastICA is very efficient however their structure search involves repeated inversions of matrices of sizes $d^2 \times d^2$ which can make it prohibitive for large problems. More explicitly, the computational complexity of LiNGAM is roughly $\mathcal{O}(N_{\mathrm{fit}} d^6)$ where $N_{\mathrm{fit}}$ is the number of model fit evaluations. In contrast, the complexity in our case is $\mathcal{O}(N_{\mathrm{ite}} d^2 N)$ where $N_{\mathrm{ite}}$ is the total number of samples including burn-in periods for both, factor model and DAG inferences. Finally, our model is more principled in the sense that all the approach is within the same Bayesian framework, as a result it can be extended to for example binary data or time series by selecting some suitable prior distributions.

Much work on Bayesian models for DAG learning already exist. For example, the approach presented in [16] is a Gaussian Bayesian network and therefore suffers from lack of identifiability. Besides, order search is performed directly for the DAG model making necessary the use of longer

sampler runs with a number of computational tricks when the problem is large ($d > 10$), i.e. when exhaustive order enumeration is not an option.

## 7 Experiments

We consider four sets of experiments in the following. The first two consist on extensive experiments using artificial data, the third addresses the model comparison scenario and the last one uses real data previously published in [17]. In every case we ran 2000 samples after a burn-in period of 4000 iterations and three independent chains for the factor model, and a single chain with 1000 samples and 2000 as burn-in for the DAG[2]. Hyperparameter settings are discussed in Section 3.

**LiNGAM suite**  We evaluate the performance of our model against LiNGAM[3] using the artificial model generator presented in [3]. The generator produces both dense and sparse networks with different degree of sparsity, $\mathbf{Z}$ is generated from a non-Gaussian heavy-tailed distribution, $\mathbf{X}$ is generated using equation (1) and then randomly permuted to hide the correct order, $\mathbf{P}$. For the experiment we have generated 1000 different dataset/models using $d = \{5, 10\}$, $N = \{200, 500, 1000, 2000\}$ and the DAG was selected using the (training set) likelihood in equation (7). Results are summarized in Figure 2 using several performance measures. For the particular case of the area under the ROC curve (AUC), we use the conditional posterior of the masking matrix, i.e. $p(\mathbf{R}|\mathbf{X}, \cdot)$ where $\mathbf{R}$ is a matrix with elements $r_{ij}$. AUC is an important measure because it quantifies how the model accounts for the uncertainty of presence or absence of links in the DAG. Such uncertainty assessment is not possible in LiNGAM where the probability of having a link is simply zero or one, however the AUC can be still computed.

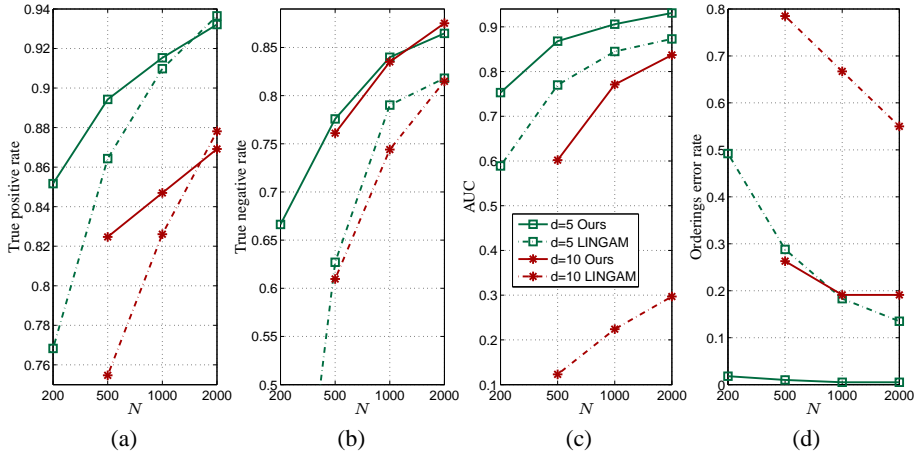

Figure 2: Performance measures for LiNGAM suite. Symbols are: square for 5 variables, star for 10 variables, solid line for sFA and dashed line for LiNGAM. (a) True positive rate. (b) True negative rate. (c) Frequency of AUC being greater than 0.9. (d) Number of estimated correct orderings.

In terms of true negative rates, AUC and ordering error rate, our approach is significantly better than LiNGAM. The true positive rate results in Figure 2(a) show that LiNGAM outperform our approach only for $N = 2000$. However by comparing it to the true positive rate, it seems than LiNGAM prefer more dense models which could be an indication of overfitting. Looking to the ordering errors, our model is clearly superior. It is important to mention that being able to compute a probability for a link in the DAG to be zero, $p(b_{ij} \neq 0|\mathbf{X}, \cdot)$, turns out to be very useful in practice, for example to reject links with high uncertainty or to rank them. To give an idea of running times on a regular two-core 2.5GHz machine, for $d = 10$ and $N = 500$: LiNGAM took in average 10 seconds and our method 170 seconds. However, when doubling the number of variables the times were 730 and 550 seconds for LiNGAM and our method respectively, which is in agreement with our complexity estimates.

**Bayesian networks repository** Next we want to compare some of the state of the art (Gaussian) approaches to DAG learning on 7 well known structures[4], namely alarm, barley, carpo, hailfinder, insurance, mildew and water ($d = 37, 48, 61, 56, 27, 35, 32$ respectively). A single dataset of size 1000 per structure was generated using a similar procedure to the one used before. Apart from ours (sFA), we considered the following methods[5]: standard DAG search (DS), order-search (OS), sparse candidate pruning then DAG-search (DSC) [6], L1MB then DAG-search (DSL) [8], sparse-candidate pruning then order-search (OSC) [7]. Results are shown in Figure 3, including the number of reversed links found due to ordering errors.

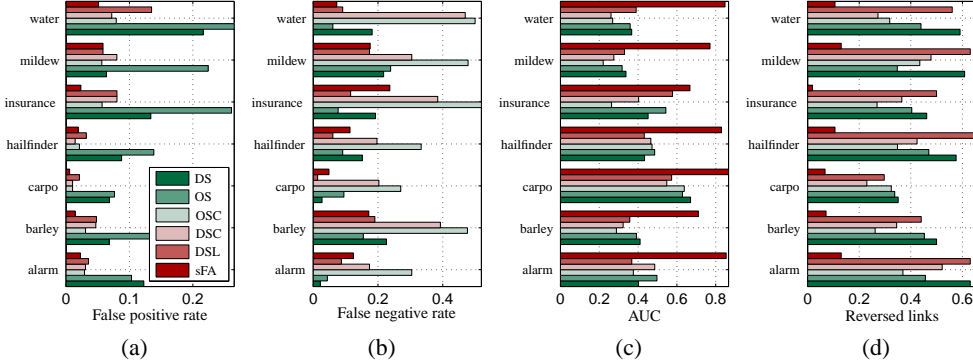

Figure 3: Performance measures for Bayesian networks repository experiments.

In this case, our approach obtained slightly better results when looking at the false positive rate, Figure 3(a). The true negative rate is comparable to the other methods suggesting that our model in some cases is sparser than the others. AUC estimates are significantly better because we have continuous probabilities for links to be zero (in the other methods we had to use a binary value). From Figure 3(d), the number of reversed links in the other methods is quite high as expected due to lack of identifiability. Our model produced a small amount reversed links because it was not able to find any of the true orderings, but indeed something quite close. This results could be improved by running the sampler for a longer time or by considering more candidates. We also tried to run the other approaches with data generated from Gaussian distributions but the results were approximately equal to those shown in Figure 3. On the other hand, our approach performs similarly but the number of reversed links increases significantly since the model is no longer identified. The most important advantage of the (Gaussian) methods used in this experiment is their speed. In all cases they are considerably faster than sampling based methods. Their speed make them very suitable for large scale problems regardless of their identifiability issues.

**Model comparison** For this experiment we have generated 1000 different datasets/models with $d = 5$ and $N = \{500, 1000\}$ in a similar way to the first experiment but this time we selected the true model to be a factor model or a DAG uniformly. In order to generate a factor model we basically just need to be sure that **A** cannot be permuted to a triangular form. We kept 20% of the data to compute the predictive densities to then select between all estimated DAG candidates and the factor model. We found that for $N = 500$ our approach was able to select true DAGs 91.5% of the times and true factor models 89.2%, corresponding to an overall error of 9.6%, For $N = 1000$ the true DAG and true factor model rates increased to 98.5% and 94.6% respectively. This results demonstrate that our approach is very effective at selecting the true underlying structure in the data between the two proposed hypotheses.

**Protein-signaling network** The dataset introduced in [17] consists on flow cytometry measurements of 11 phosphorylated proteins and phospholipids (Raf, Erk, p38, Jnk, Akt, Mek, PKA, PKC, $PIP_2$, $PIP_3$, $PLC\gamma$). Each observation is a vector of quantitative amounts measured from single cells, generated from a series of stimulatory cues and inhibitory interventions. The dataset contains both observational and experimental data. Here we are only using 1755 samples corresponding to

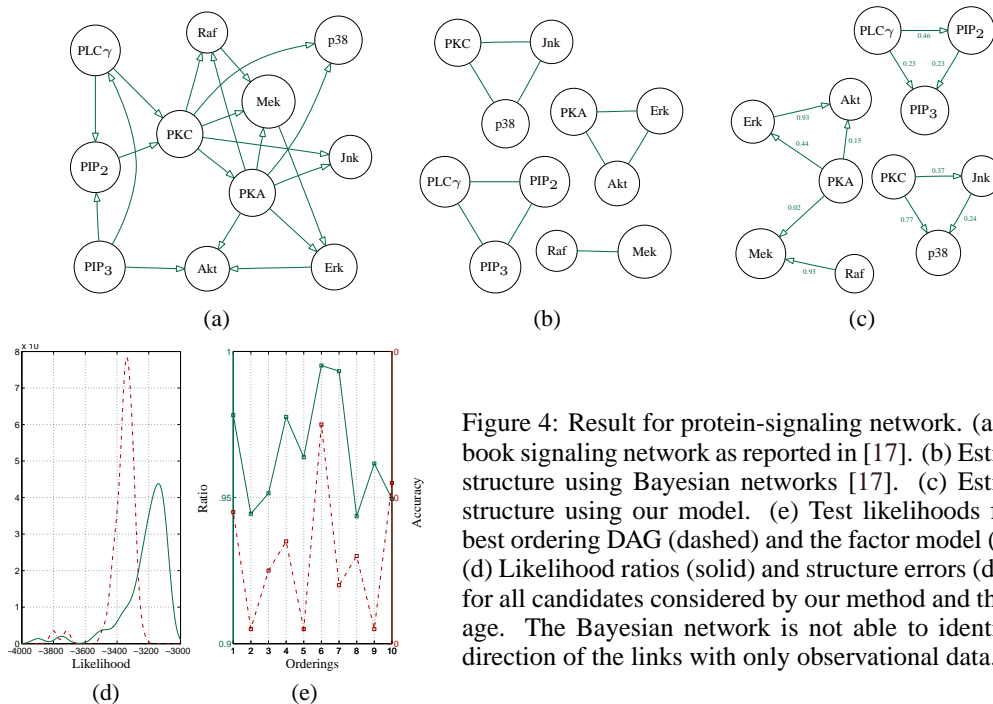

Figure 4: Result for protein-signaling network. (a) Textbook signaling network as reported in [17]. (b) Estimated structure using Bayesian networks [17]. (c) Estimated structure using our model. (e) Test likelihoods for the best ordering DAG (dashed) and the factor model (solid). (d) Likelihood ratios (solid) and structure errors (dashed) for all candidates considered by our method and their usage. The Bayesian network is not able to identify the direction of the links with only observational data.

pure observational data and randomly selected $20\%$ of the data to compute the predictive densities. Using the entire set will produce a richer model, however interventions are out of the scope of this paper. The textbook ground truth and results are presented in figure 4. From the 21 possible links in figure 4(a), the model from [17] was able to find 9, but also one falsely added link. In 4(b), a marginal likelihood equivalent prior is used and they therefore cannot make any inferences about directionality from observational data alone, see Figure 4(b). Our model in Figure 4(c) was able to find 10 true links, one falsely added link and only two reversed links (RL), one of them is $PIP_2 \rightarrow PIP_3$ which according to the ground truth is bidirectional and the other one, $PLC\gamma \rightarrow PIP_3$ which was also found reversed using experimental data in [17]. Note from figure 4(e) that the predictive density ratios correlate quite well with the structural accuracy. The predictive densities for the best candidate (sixth in Figure 4(e)) is shown in Figure 4(d) and suggests that the factor model is a better option which makes sense considering that estimated DAG in figure 4(c) is a substructure of the ground truth. We also examined the estimated factor model and we found out that three factors could correspond to unmeasured proteins (PI3K, MKK and IP3), see Figure 2 and table 3 in [17]. We also tried the above methods. Results were very similar to our method in terms of true positives ($\approx 9$) and true negatives ($\approx 32$), however none of them were able to produce less than 6 reversed links that corresponds to approximately two-thirds of total true positives.

## 8 Discussion

We have proposed a novel approach to perform inference and model comparison of sparse factor models and DAGs within the same framework. The key ingredients for both Bayesian models are spike and slab priors to promote sparsity, heavy-tailed priors to ensure identifiability and predictive densities to perform the comparison. A set of candidate orderings is produced by the factor model. Subsequently, a linear DAG is learned for each of the candidates. To the authors' knowledge this is the first time that a method for comparing such a closely related linear models is proposed. This setting can be very beneficial in situations where the prior evidence suggests both DAG structure and/or unmeasured variables in the data. For example in the protein signaling network [17], the textbook ground truth suggests both DAG structure and a number of unmeasured proteins. The previous approach [17] only performed structure learning in DAGs but our results suggest that the data is better explained by the factor model. For further exploration of this data set, we obviously need to modify our approach to handle hybrid models, i.e. graphs with directed/undirected links and observed/latent nodes as well as being able to use experimental data. Our Bayesian hierarchical approach is very flexible. We are currently investigating extensions to other source distributions (non-parametric Dirichlet process, temporal Gaussian processes and discrete).

## Footnotes

[1]These ambiguities are not affecting our ability to find correct permutation $\mathbf{P}$ of the rows.

[2]Source code available upon request (C with Matlab interface).

[3]Matlab package available at http://www.cs.helsinki.fi/group/neuroinf/lingam/.

[4]http://compbio.cs.huji.ac.il/Repository/.

[5]Parameters: 10000 iterations, 5 candidates (SC, DSC), max fan-in of 5 (OS, OSC) and *Or* strategy and MDL penalty (DSL).

# References

[1] M. West. Bayesian factor regression models in the "large $p$, small $n$" paradigm. In J. Bernardo, M. Bayarri, J. Berger, A. Dawid, D. Heckerman, A. Smith, and M. West, editors, *Bayesian Statistics 7*, pages 723–732. Oxford University Press, 2003.

[2] J. Lucas, C. Carvalho, Q. Wang, A. Bild, J. R. Nevins, and M. West. *Bayesian Inference for Gene Expression and Proteomics*, chapter Sparse Statistical Modeling in Gene Expression Genomics, pages 155–176. Cambridge University Press, 2006.

[3] S. Shimizu, P. O. Hoyer, A. Hyvärinen, and A. Kerminen. A linear non-Gaussian acyclic model for causal discovery. *Journal of Machine Learning Research*, 7:2003–2030, October 2006.

[4] D. M. Chickering. Learning Bayesian networks is NP-complete. In D. Fisher and H.-J. Lenz, editors, *Learning from Data: AI and Statistics*, pages 121–130. Springer-Verlag, 1996.

[5] I. Tsamardinos, L. E. Brown, and C. F. Aliferis. The max-min hill-climbing Bayesian network structure learning algorithm. *Machine Learning*, 65(1):31–78, October 2006.

[6] N. Friedman, I. Nachman, and D. Pe'er. Learning Bayesian network structure from massive datasets: The "sparse candidate" algorithm. In K. B. Laskey and H. Prade, editors, *UAI*, pages 206–215, 1999.

[7] M. Teyssier and D. Koller. Ordering-based search: A simple and effective algorithm for learning Bayesian networks. In *UAI*, pages 548–549, 2005.

[8] M. W. Schmidt, A. Niculescu-Mizil, and K. P. Murphy. Learning graphical model structure using L1-regularization paths. In *AAAI*, pages 1278–1283, 2007.

[9] D. Heckerman, D. Geiger, and D. M. Chickering. Learning Bayesian networks: The combination of knowledge and statistical data. *Machine Learning*, 20(3):197–243, January 1995.

[10] J. Pearl. *Causality: Models, Reasoning, and Inference*. Cambridge University Press, March 2000.

[11] P. Comon. Independent component analysis, a new concept? *Signal Processing*, 36(3):287–314, December 1994.

[12] C. M. Carvalho, J. Chang, J. E. Lucas, J. R. Nevins, Q. Wang, and M. West. High-dimensional sparse factor modeling: Applications in gene expression genomics. *Journal of the American Statistical Association*, 103(484):1438–1456, December 2008.

[13] A. Hyvärinen, J. Karhunen, and E. Oja. *Independent Component Analysis*. Wiley-Interscience, May 2001.

[14] D. F. Andrews and C. L. Mallows. Scale mixtures of normal distributions. *Journal of the Royal Statistical Society: Series B (Methodology)*, 36(1):99–102, 1974.

[15] T. Park and G. Casella. The Bayesian lasso. *Journal of the American Statistical Association*, 103(482):681–686, June 2008.

[16] N. Friedman and D. Koller. Being Bayesian about network structure: A Bayesian approach to structure discovery in Bayesian networks. *Machine Learning*, 50(1–2):95–125, January 2003.

[17] K. Sachs, O. Perez, D. Pe'er, D. A. Lauffenburger, and G. P. Nolan. Causal protein-signaling networks derived from multiparameter single-cell data. *Science*, 308(5721):523–529, April 2005.

